# Direct Loss Minimization for Structured Prediction

**David McAllester**
TTI-Chicago
mcallester@ttic.edu

**Tamir Hazan**
TTI-Chicago
tamir@ttic.edu

**Joseph Keshet**
TTI-Chicago
jkeshet@ttic.edu

## Abstract

In discriminative machine learning one is interested in training a system to optimize a certain desired measure of performance, or loss. In binary classification one typically tries to minimizes the error rate. But in structured prediction each task often has its own measure of performance such as the BLEU score in machine translation or the intersection-over-union score in PASCAL segmentation. The most common approaches to structured prediction, structural SVMs and CRFs, do not minimize the task loss: the former minimizes a surrogate loss with no guarantees for task loss and the latter minimizes log loss independent of task loss. The main contribution of this paper is a theorem stating that a certain perceptron-like learning rule, involving features vectors derived from loss-adjusted inference, directly corresponds to the gradient of task loss. We give empirical results on phonetic alignment of a standard test set from the TIMIT corpus, which surpasses all previously reported results on this problem.

## 1 Introduction

Many modern software systems compute a result as the solution, or approximate solution, to an optimization problem. For example, modern machine translation systems convert an input word string into an output word string in a different language by approximately optimizing a score defined on the input-output pair. Optimization underlies the leading approaches in a wide variety of computational problems including problems in computational linguistics, computer vision, genome annotation, advertisement placement, and speech recognition. In many optimization-based software systems one must design the objective function as well as the optimization algorithm. Here we consider a parameterized objective function and the problem of setting the parameters of the objective in such a way that the resulting optimization-driven software system performs well.

We can formulate an abstract problem by letting $\mathcal{X}$ be an abstract set of possible inputs and $\mathcal{Y}$ an abstract set of possible outputs. We assume an objective function $s_w : \mathcal{X} \times \mathcal{Y} \to \mathbb{R}$ parameterized by a vector $w \in \mathbb{R}^d$ such that for $x \in \mathcal{X}$ and $y \in \mathcal{Y}$ we have a score $s_w(x, y)$. The parameter setting $w$ determines a mapping from input $x$ to output $y_w(x)$ is defined as follows:

$$y_w(x) = \operatorname*{argmax}_{y \in \mathcal{Y}} s_w(x, y) \tag{1}$$

Our goal is to set the parameters $w$ of the scoring function such that the mapping from input to output defined by (1) performs well. More formally, we assume that there exists some unknown probability distribution $\rho$ over pairs $(x, y)$ where $y$ is the desired output (or reference output) for input $x$. We assume a loss function $L$, such as the BLEU score, which gives a cost $L(y, \hat{y}) \geq 0$ for producing output $\hat{y}$ when the desired output (reference output) is $y$. We then want to set $w$ so as to minimize the expected loss.

$$w^* = \operatorname*{argmin}_{w} \mathbb{E}\left[L(y, y_w(x))\right] \tag{2}$$

In (2) the expectation is taken over a random draw of the pair $(x, y)$ form the source data distribution $\rho$. Throughout this paper all expectations will be over a random draw of a fresh pair $(x, y)$. In machine learning terminology we refer to (1) as inference and (2) as training.

Unfortunately the training objective function (2) is typically non-convex and we are not aware of any polynomial algorithms (in time and sample complexity) with reasonable approximation guarantees to (2) for typical loss functions, say 0-1 loss, and an arbitrary distribution $\rho$. In spite of the lack of approximation guarantees, it is common to replace the objective in (2) with a convex relaxation such as structural hinge loss [8, 10]. It should be noted that replacing the objective in (2) with structural hinge loss leads to inconsistency — the optimum of the relaxation is different from the optimum of (2).

An alternative to a convex relaxation is to perform gradient descent directly on the objective in (2). In some applications it seems possible that the local minima problem of non-convex optimization is less serious than the inconsistencies introduced by a convex relaxation.

Unfortunately, direct gradient descent on (2) is conceptually puzzling in the case where the output space $\mathcal{Y}$ is discrete. In this case the output $y_w(x)$ is not a differentiable function of $w$. As one smoothly changes $w$ the output $y_w(x)$ jumps discontinuously between discrete output values. So one cannot write $\nabla_w \mathbb{E}\left[L(y, y_w(x))\right]$ as $\mathbb{E}\left[\nabla_w L(y, y_w(x))\right]$. However, when the input space $\mathcal{X}$ is continuous the gradient $\nabla_w \mathbb{E}\left[L(y, y_w(x))\right]$ can exist even when the output space $\mathcal{Y}$ is discrete. The main results of this paper is a perceptron-like method of performing direct gradient descent on (2) in the case where the output space is discrete but the input space is continuous.

After formulating our method we discovered that closely related methods have recently become popular for training machine translation systems [7, 2]. Although machine translation has discrete inputs as well as discrete outputs, the training method we propose can still be used, although without theoretical guarantees. We also present empirical results on the use of this method in phoneme alignment on the TIMIT corpus, where it achieves the best known results on this problem.

## 2 Perceptron-Like Training Methods

Perceptron-like training methods are generally formulated for the case where the scoring function is linear in $w$. In other words, we assume that the scoring function can be written as follows where $\phi : \mathcal{X} \times \mathcal{Y} \to \mathbb{R}^d$ is called a *feature map*.

$$s_w(x, y) = w^\top \phi(x, y)$$

Because the feature map $\phi$ can itself be nonlinear, and the feature vector $\phi(x, y)$ can be very high dimensional, objective functions of the this form are highly expressive.

Here we will formulate perceptron-like training in the data-rich regime where we have access to an unbounded sequence $(x_1, y_1)$, $(x_2, y_2)$, $(x_3, y_3)$, ... where each $(x_t, y_t)$ is drawn IID from the distribution $\rho$. In the basic structured prediction perceptron algorithm [3] one constructs a sequence of parameter settings $w^0$, $w^1$, $w^2$, ... where $w^0 = 0$ and $w^{t+1}$ is defined as follows.

$$w^{t+1} = w^t + \phi(x_t, y_t) - \phi(x_t, y_{w^t}(x_t)) \tag{3}$$

Note that if $y_{w^t}(x_t) = y_t$ then no update is made and we have $w^{t+1} = w^t$. If $y_{w^t}(x_t) \neq y_t$ then the update changes the parameter vector in a way that favors $y_t$ over $y_{w^t}(x_t)$. If the source distribution $\rho$ is $\gamma$-separable, i.e., there exists a weight vector $w$ with the property that $y_w(x) = y$ with probability 1 and $y_w(x)$ is always $\gamma$-separated from all distractors, then the perceptron update rule will eventually lead to a parameter setting with zero loss. Note, however, that the basic perceptron update does not involve the loss function $L$. Hence it cannot be expected to optimize the training objective (2) in cases where zero loss is unachievable.

A loss-sensitive perceptron-like algorithm can be derived from the structural hinge loss of a margin-scaled structural SVM [10]. The optimization problem for margin-scaled structural hinge loss can be defined as follows.

$$w^* = \underset{w}{\operatorname{argmin}} \, \mathbb{E}\left[\max_{\tilde{y} \in \mathcal{Y}} \left(L(y, \tilde{y}) - w^\top \left(\phi(x, y) - \phi(x, \tilde{y})\right)\right)\right]$$

It can be shown that this is a convex relaxation of (2). We can optimize this convex relaxation with stochastic sub-gradient descent. To do this we compute a sub-gradient of the objective by first computing the value of $\tilde{y}$ which achieves the maximum.

$$
\begin{aligned}
y_{hinge}^t &= \underset{\tilde{y} \in \mathcal{Y}}{\operatorname{argmax}} \, L(y_t, \tilde{y}) - (w^t)^\top \left(\phi(x_t, y_t) - \phi(x_t, \tilde{y})\right) \\
&= \underset{\tilde{y} \in \mathcal{Y}}{\operatorname{argmax}} \, (w^t)^\top \phi(x_t, \tilde{y}) + L(y_t, \tilde{y}) \tag{4}
\end{aligned}
$$

This yields the following perceptron-like update rule where the update direction is the negative of the sub-gradient of the loss and $\eta^t$ is a learning rate.

$$w^{t+1} = w^t + \eta^t \left( \phi(x_t, y_t) - \phi(x_t, y_{\text{hinge}}^t) \right) \tag{5}$$

Equation (4) is often referred to as *loss-adjusted inference*. The use of loss-adjusted inference causes the rule update (5) to be at least influenced by the loss function.

Here we consider the following perceptron-like update rule where $\eta^t$ is a time-varying learning rate and $\epsilon^t$ is a time-varying loss-adjustment weight.

$$w^{t+1} = w^t + \eta^t \left( \phi(x_t, y_{w^t}(x_t)) - \phi(x_t, y_{\text{direct}}^t) \right) \tag{6}$$

$$y_{\text{direct}}^t = \operatorname*{argmax}_{\tilde{y} \in \mathcal{Y}} (w^t)^\top \phi(x_t, \tilde{y}) + \epsilon^t L(y, \tilde{y}) \tag{7}$$

In the update (6) we view $y_{\text{direct}}^t$ as being worse than $y_{w^t}(x_t)$. The update direction moves away from feature vectors of larger-loss labels. Note that the reference label $y_t$ in (5) has been replaced by the inferred label $y_{w^t}(x)$ in (6). The main result of this paper is that under mild conditions the expected update direction of (6) approaches the negative direction of $\nabla_w \mathbb{E}\left[L(y, y_w(x))\right]$ in the limit as the update weight $\epsilon^t$ goes to zero. In practice we use a different version of the update rule which moves toward better labels rather than away from worse labels. The toward-better version is given in Section 5. Our main theorem applies equally to the toward-better and away-from-worse versions of the rule.

## 3 The Loss Gradient Theorem

The main result of this paper is the following theorem.

**Theorem 1.** *For a finite set $\mathcal{Y}$ of possible output values, and for $w$ in general position as defined below, we have the following where $y_{\text{direct}}$ is a function of $w$, $x$, $y$ and $\epsilon$.*

$$\nabla_w \mathbb{E}\left[L(y, y_w(x))\right] = \lim_{\epsilon \to 0} \frac{1}{\epsilon} \mathbb{E}\left[\phi(x, y_{\text{direct}}) - \phi(x, y_w(x)))\right]$$

*where*

$$y_{\text{direct}} = \operatorname*{argmax}_{\tilde{y} \in \mathcal{Y}} w^\top \phi(x, \tilde{y}) + \epsilon L(y, \tilde{y})$$

We prove this theorem in the case of only two labels where we have $y \in \{-1, 1\}$. Although the proof is extended to the general case in a straight forward manner, we omit the general case to maintain the clarity of the presentation. We assume an input set $\mathcal{X}$ and a probability distribution or a measure $\rho$ on $\mathcal{X} \times \{-1, 1\}$ and a loss function $L(y, y')$ for $y, y' \in \{-1, 1\}$. Typically the loss $L(y, y')$ is zero if $y = y'$ but the loss of a false positive, namely $L(-1, 1)$, may be different from the loss of a false negative, $L(1, -1)$.

By definition the gradient of expected loss satisfies the following condition for any vector $\Delta w \in \mathbb{R}^d$.

$$\Delta w^\top \nabla_w \mathbb{E}\left[L(y, y_w(x))\right] = \lim_{\epsilon \to 0} \frac{\mathbb{E}\left[L(y, y_{w + \epsilon \Delta w}(x))\right] - \mathbb{E}\left[L(y, y_w(x))\right]}{\epsilon}$$

Using this observation, the direct loss theorem is equivalent to the following

$$\lim_{\epsilon \to 0} \frac{\mathbb{E}\left[L(y, y_{w + \epsilon \Delta w}(x)) - L(y, y_w(x))\right]}{\epsilon} = \lim_{\epsilon \to 0} \frac{(\Delta w)^\top \mathbb{E}\left[\phi(x, y_{\text{direct}}) - \phi(x, y_w(x))\right]}{\epsilon} \tag{8}$$

For the binary case we define $\Delta \phi(x) = \phi(x, 1) - \phi(x, -1)$. Under this convention we have $y_w(x) = \text{sign}(w^\top \Delta \phi(x))$. We first focus on the left hand side of (8). If the two labels $y_{w+\epsilon \Delta w}(x)$ and $y_w(x)$ are the same then the quantity inside the expectation is zero. We now define the following two sets which correspond to the set of inputs $x$ for which these two labels are different.

$$
\begin{aligned}
S_\epsilon^+ &= \{x : y_w(x) = -1, \ y_{w+\epsilon \Delta w}(x) = 1\} \\
&= \{x : w^\top \Delta \phi(x) < 0, \ (w + \epsilon \Delta w)^\top \Delta \phi(x) \geq 0\} \\
&= \{x : w^\top \Delta \phi(x) \in [-\epsilon (\Delta w)^\top \Delta \phi(x), 0)\}
\end{aligned}
$$

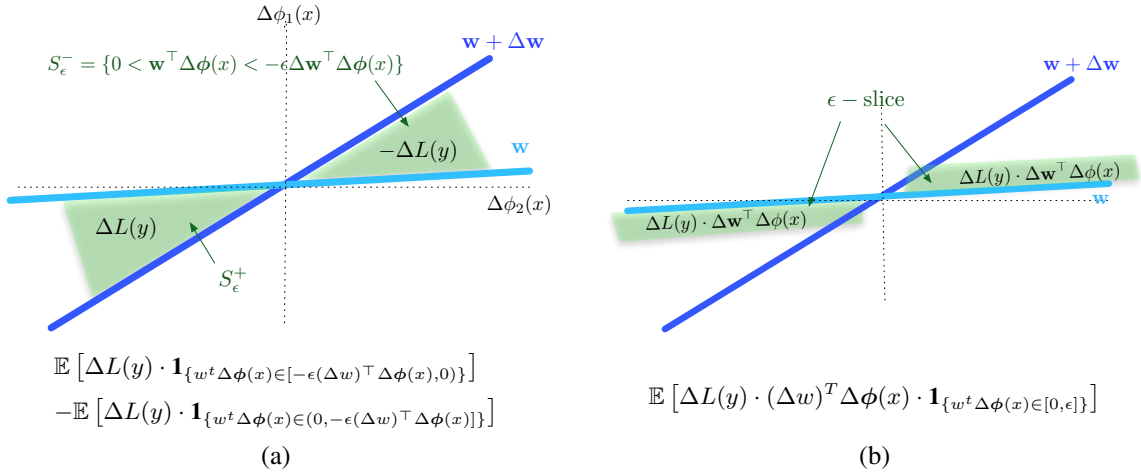

$$\mathbb{E}\left[\Delta L(y) \cdot \mathbf{1}_{\{w^t \Delta\phi(x) \in [-\epsilon(\Delta w)^\top \Delta\phi(x),0)\}}\right]$$
$$-\mathbb{E}\left[\Delta L(y) \cdot \mathbf{1}_{\{w^t \Delta\phi(x) \in (0,-\epsilon(\Delta w)^\top \Delta\phi(x)]\}}\right]$$

(a)

$$\mathbb{E}\left[\Delta L(y) \cdot (\Delta w)^T \Delta\phi(x) \cdot \mathbf{1}_{\{w^t \Delta\phi(x) \in [0,\epsilon]\}}\right]$$

(b)

Figure 1: Geometrical interpretation of the loss gradient. In (a) we illustrate the integration of the constant value $\Delta L(y)$ over the set $S_\epsilon^+$ and the constant value $-\Delta L(y)$ over the set $S_\epsilon^-$ (the green area). The lines represent the decision boundaries defined by the associated vectors. In (b) we show the integration of $\Delta L(y)(\Delta w)^\top \Delta\phi(x)$ over the sets $U_\epsilon^+ = \{x : w^t \Delta\phi(x) \in [0,\epsilon]\}$ and $U_\epsilon^- = \{x : w^t \Delta\phi(x) \in [-\epsilon(\Delta w)^\top \Delta\phi(x),0)\}$. The key observation of the proof is that under very general conditions these integrals are asymptotically equivalent in the limit as $\epsilon$ goes to zero.

and

$$
\begin{aligned}
S_\epsilon^- &= \{x : y_w(x) = 1, \ y_{w+\epsilon\Delta w}(x) = -1\} \\
&= \{x : w^\top \Delta\phi(x) \ge 0, \ (w+\epsilon\Delta w)^\top \Delta\phi(x) < 0\} \\
&= \{x : w^\top \Delta\phi(x) \in [0, -\epsilon(\Delta w)^\top \Delta\phi(x))\}
\end{aligned}
$$

We define $\Delta L(y) = L(y,1) - L(y,-1)$ and then write the left hand side of (8) as follows.

$$\mathbb{E}\left[L(y, y_{w+\epsilon\Delta w}(x)) - L(y, y_w(x))\right] = \mathbb{E}\left[\Delta L(y)\mathbf{1}_{\{x \in S_\epsilon^+\}}\right] - \mathbb{E}\left[\Delta L(y)\mathbf{1}_{\{x \in S_\epsilon^-\}}\right] \qquad (9)$$

These expectations are shown as integrals in Figure 1 (a) where the lines in the figure represent the decision boundaries defined by $w$ and $w + \epsilon\Delta w$.

To analyze this further we use the following lemma.

**Lemma 1.** *Let $Z(z)$, $U(u)$ and $V(v)$ be three real-valued random variables whose joint measure $\rho$ can be expressed as a measure $\mu$ on $U$ and $V$ and a bounded continuous conditional density function $f(z|u,v)$. More rigorously, we require that for any $\rho$-measurable set $S \subseteq \mathbb{R}^3$ we have $\rho(S) = \int_{u,v}\left[\int_z f(z|u,v)\mathbf{1}_{\{z,u,v\in S\}}dz\right]d\mu(u,v)$. For any three such random variables we have the following.*

$$
\begin{aligned}
\lim_{\epsilon \to +0} \frac{1}{\epsilon}\left(\mathbb{E}_\rho\left[U \cdot \mathbf{1}_{\{z\in[0,\epsilon V]\}}\right] - \mathbb{E}_\rho\left[U \cdot \mathbf{1}_{\{z\in[\epsilon V,0]\}}\right]\right) &= \mathbb{E}_\mu\left[UV \cdot f(0|u,v)\right] \\
&= \lim_{\epsilon \to +0} \frac{1}{\epsilon}\mathbb{E}_\rho\left[UV \cdot \mathbf{1}_{\{z\in[0,\epsilon]\}}\right]
\end{aligned}
$$

*Proof.* First we note the following where $V^+$ denotes $\max(0, V)$.

$$
\begin{aligned}
\lim_{\epsilon \to +0} \frac{1}{\epsilon}\mathbb{E}_\rho\left[U \cdot \mathbf{1}_{\{z\in[0,\epsilon V)\}}\right] &= \lim_{\epsilon \to +0} \frac{1}{\epsilon}\mathbb{E}_\mu\left[U\int_0^{\epsilon V} f(z|u,v)dz\right] \\
&= \mathbb{E}_\mu\left[UV^+ \cdot f(0|U,V)\right]
\end{aligned}
$$

Similarly we have the following where $V^-$ denotes $\min(0, V)$.

$$
\begin{aligned}
\lim_{\epsilon \to +0} \frac{1}{\epsilon}\mathbb{E}_\rho\left[U \cdot \mathbf{1}_{\{z\in(\epsilon V,0]\}}\right] &= \lim_{\epsilon \to +0} \frac{1}{\epsilon}\mathbb{E}_\mu\left[U\int_{\epsilon V}^0 f(z|u,v)dz\right] \\
&= -\mathbb{E}_\mu\left[UV^- \cdot f(0|U,V)\right]
\end{aligned}
$$

Subtracting these two expressions gives the following.

$$\mathbb{E}_\mu\left[UV^+ \cdot f(0|U,V)\right] + \mathbb{E}_\mu\left[UV^- \cdot f(0|U,V)\right] \;=\; \mathbb{E}_\mu\left[U(V^+ + V^-) \cdot f(0|U,V)\right]$$
$$=\; \mathbb{E}_\mu\left[UV \cdot f(0|U,V)\right]$$

$\square$

Applying Lemma 1 to (9) with $Z$ being the random variable $w^T\Delta\phi(x)$, $U$ being the random variable $-\Delta L(y)$ and $V$ being $-(\Delta w)^T\Delta\phi(x)$ yields the following.

$$(\Delta w)^\top \nabla_w \mathbb{E}\left[L(y, y_w(x))\right] \;=\; \lim_{\epsilon\to^+ 0}\frac{1}{\epsilon}\mathbb{E}\left[\Delta L(y)\cdot\mathbf{1}_{\{x\in S_\epsilon^+\}}\right] - \mathbb{E}\left[\Delta L(y)\cdot\mathbf{1}_{\{x\in S_\epsilon^-\}}\right]$$
$$=\; \lim_{\epsilon\to^+ 0}\frac{1}{\epsilon}\mathbb{E}\left[\Delta L(y)\cdot(\Delta w)^\top\Delta\phi(x)\cdot\mathbf{1}_{\{w^\top\Delta\phi\in[0,\epsilon]\}}\right] \quad (10)$$

Of course we need to check that the conditions of Lemma 1 hold. This is where we need a general position assumption for $w$. We discuss this issue briefly in Section 3.1.

Next we consider the right hand side of (8). If the two labels $y_{\text{direct}}$ and $y_w(x)$ are the same then the quantity inside the expectation is zero. We note that we can write $y_{\text{direct}}$ as follows.

$$y_{\text{direct}} = \text{sign}\left(w^\top\Delta\phi(x) + \epsilon\Delta L(y)\right)$$

We now define the following two sets which correspond to the set of pairs $(x,y)$ for which $y_w(x)$ and $y_{\text{direct}}$ are different.

$$B_\epsilon^+ \;=\; \{(x,y) : y_w(x) = -1, \; y_{\text{direct}} = 1\}$$
$$=\; \{(x,y) : w^\top\Delta\phi(x) < 0, \; w^\top\Delta\phi(x) + \epsilon\Delta L(y) \geq 0\}$$
$$=\; \{(x,y) : w^\top\Delta\phi(x) \in [-\epsilon\Delta L(y)(x), 0)\}$$

$$B_\epsilon^- \;=\; \{(x,y) : y_w(x) = 1, \; y_{\text{direct}} = -1\}$$
$$=\; \{(x,y) : w^\top\Delta\phi(x) \geq 0, \; w^\top\Delta\phi(x) + \epsilon\Delta L(y) < 0\}$$
$$=\; \{(x,y) : w^\top\Delta\phi(x) \in [0, -\epsilon\Delta L(y))\}$$

We now have the following.

$$\mathbb{E}\left[(\Delta w)^\top\left(\phi(x, y_{\text{direct}}) - \phi(x, y_w(x))\right)\right]$$
$$=\; \mathbb{E}\left[(\Delta w)^\top\Delta\phi(x)\cdot\mathbf{1}_{\{(x,y)\in B_\epsilon^+\}}\right] - \mathbb{E}\left[(\Delta w)^\top\Delta\phi(x)\cdot\mathbf{1}_{\{(x,y)\in B_\epsilon^-\}}\right] \quad (11)$$

These expectations are shown as integrals in Figure 1 (b). Applying Lemma 1 to (11) with $Z$ set to $w^\top\Delta\phi(x)$, $U$ set to $-(\Delta w)^\top\Delta\phi(x)$ and $V$ set to $-\Delta L(y)$ gives the following.

$$\lim_{\epsilon\to^+ 0}\frac{1}{\epsilon}(\Delta w)^\top\mathbb{E}\left[\phi(x, y_{\text{direct}}) - \phi(x, y_w(x))\right]$$
$$=\; \lim_{\epsilon\to^+ 0}\frac{1}{\epsilon}\mathbb{E}\left[(\Delta w)^\top\Delta\phi(x)\cdot\Delta L(y)\cdot\mathbf{1}_{\{w^\top\Delta\phi(x)\in[0,\epsilon]\}}\right] \quad (12)$$

Theorem 1 now follows from (10) and (12).

## 3.1  The General Position Assumption

The general position assumption is needed to ensure that Lemma 1 can be applied in the proof of Theorem 1. As a general position assumption, it is sufficient, but not necessary, that $w \neq 0$ and $\phi(x,y)$ has a bounded density on $\mathbb{R}^d$ for each fixed value of $y$. It is also sufficient that the range of the feature map is a submanifold of $\mathbb{R}^d$ and $\phi(x,y)$ has a bounded density relative to the surface of that submanifold, for each fixed value of $y$. More complex distributions and feature maps are also possible.

# 4 Extensions: Approximate Inference and Latent Structure

In many applications the inference problem (1) is intractable. Most commonly we have some form of graphical model. In this case the score $w^\top \phi(x,y)$ is defined as the negative energy of a Markov random field (MRF) where $x$ and $y$ are assignments of values to nodes of the field. Finding a lowest energy value for $y$ in (1) in a general graphical model is NP-hard.

A common approach to an intractable optimization problem is to define a convex relaxation of the objective function. In the case of graphical models this can be done by defining a relaxation of a marginal polytope [11]. The details of the relaxation are not important here. At a very abstract level the resulting approximate inference problem can be defined as follows where the set $R$ is a relaxation of the set $\mathcal{Y}$, and corresponds to the extreme points of the relaxed polytope.

$$r_w(x) = \operatorname*{argmax}_{r \in \mathcal{R}} w^\top \phi(x, r) \tag{13}$$

We assume that for $y \in \mathcal{Y}$ and $r \in \mathcal{R}$ we can assign a loss $L(y, r)$. In the case of a relaxation of the marginal polytope of a graphical model we can take $L(y, r)$ to be the expectation over a random rounding of $r$ to $\tilde{y}$ of $L(y, \tilde{y})$. For many loss functions, such as weighted Hamming loss, one can compute $L(y, r)$ efficiently. The training problem is then defined by the following equation.

$$w^* = \operatorname*{argmin}_{w} \mathbb{E}\left[L(y, r_w(x))\right] \tag{14}$$

Note that (14) directly optimizes the performance of the approximate inference algorithm. The parameter setting optimizing approximate inference might be significantly different from the parameter setting optimizing the loss under exact inference.

The proof of Theorem 1 generalizes to (14) provided that $R$ is a finite set, such as the set of vertices of a relaxation of the marginal polytope. So we immediately get the following generalization of Theorem 1.

$$\nabla_w \mathbb{E}_{(x,y)\sim\rho}\left[L(y, r_w(x))\right] = \lim_{\epsilon \to 0} \frac{1}{\epsilon} \mathbb{E}\left[\phi(x, r_{\text{direct}}) - \phi(x, r_w(x))\right]$$

where

$$r_{\text{direct}} = \operatorname*{argmax}_{\tilde{r} \in \mathcal{R}} w^\top \phi(x, \tilde{r}) + \epsilon L(y, \tilde{r})$$

Another possible extension involves hidden structure. In many applications it is useful to introduce hidden information into the inference optimization problem. For example, in machine translation we might want to construct parse trees for the both the input and output sentence. In this case the inference equation can be written as follows where $h$ is the hidden information.

$$y_w(x) = \operatorname*{argmax}_{y \in \mathcal{Y}} \max_{h \in \mathcal{H}} w^\top \phi(x, y, h) \tag{15}$$

In this case we can take the training problem to again be defined by (2) but where $y_w(x)$ is defined by (15).

Latent information can be handled by the equations of approximate inference but where $R$ is reinterpreted as the set of pairs $(y, h)$ with $y \in \mathcal{Y}$ and $h \in \mathcal{H}$. In this case $L(y, r)$ has the form $L(y, (y', h))$ which we can take to be equal to $L(y, y')$.

# 5 Experiments

In this section we present empirical results on the task of phoneme-to-speech alignment. Phoneme-to-speech alignment is used as a tool in developing speech recognition and text-to-speech systems. In the phoneme alignment problem each input $\overline{x}$ represents a speech utterance, and consists of a pair $(\overline{s}, \overline{p})$ of a sequence of acoustic feature vectors, $\overline{s} = (s_1, \ldots, s_T)$, where $s_t \in \mathbb{R}^d$, $1 \leq t \leq T$; and a sequence of phonemes $\overline{p} = (p_1, \ldots, p_K)$, where $p_k \in \mathcal{P}$, $1 \leq k \leq K$ is a phoneme symbol and $\mathcal{P}$ is a finite set of phoneme symbols. The lengths $K$ and $T$ can be different for different inputs although typically we have $T$ significantly larger than $K$. The goal is to generate an alignment between the two sequences in the input. Sometimes this task is called *forced-alignment* because one is forced

Table 1: Percentage of correctly positioned phoneme boundaries, given a predefined tolerance on the TIMIT corpus. Results are reported on the whole TIMIT test-set (1344 utterances).

| | $\tau$-alignment accuracy [%] | | | | $\tau$-insensitive |
| | $t \leq 10$ms | $t \leq 20$ms | $t \leq 30$ms | $t \leq 40$ms | loss |
| --- | --- | --- | --- | --- | --- |
| Brugnara *et al.* (1993) | 74.6 | 88.8 | 94.1 | 96.8 | |
| Keshet (2007) | 80.0 | 92.3 | 96.4 | 98.2 | - |
| Hosom (2009) | 79.30 | 93.36 | 96.74 | 98.22 | - |
| Direct loss min. (trained $\tau$-alignment) | **86.01** | 94.08 | 97.08 | 98.44 | 0.278 |
| Direct loss min. (trained $\tau$-insensitive) | 85.72 | **94.21** | **97.21** | **98.60** | 0.277 |

to interpret the given acoustic signal as the given phoneme sequence. The output $\bar{y}$ is a sequence $(y_1, \ldots, y_K)$, where $1 \leq y_k \leq T$ is an integer giving the start frame in the acoustic sequence of the $k$-th phoneme in the phoneme sequence. Hence the $k$-th phoneme starts at frame $y_k$ and ends at frame $y_{k+1} - 1$.

Two types of loss functions are used to quantitatively assess alignments. The first loss is called the $\tau$-*alignment loss* and it is defined as

$$L^{\tau\text{-alignment}}(\bar{y}, \bar{y}') = \frac{1}{|\bar{y}|} \left| \{k : |y_k - y_k'| > \tau\} \right|. \tag{16}$$

In words, this loss measures the average number of times the absolute difference between the predicted alignment sequence and the manual alignment sequence is greater than $\tau$. This loss with different values of $\tau$ was used to measure the performance of the learned alignment function in [1, 9, 4]. The second loss, called $\tau$-*insensitive loss* was proposed in [5] as is defined as follows.

$$L^{\tau\text{-insensitive}}(\bar{y}, \bar{y}') = \frac{1}{|\bar{y}|} \max\{|y_k - y_k'| - \tau, 0\} \tag{17}$$

This loss measures the average disagreement between all the boundaries of the desired alignment sequence and the boundaries of predicted alignment sequence where a disagreement of less than $\tau$ is ignored. Note that $\tau$-insensitive loss is continuous and convex while $\tau$-alignment is discontinuous and non-convex. Rather than use the "away-from-worse" update given by (6) we use the "toward-better" update defined as follows. Both updates give the gradient direction in the limit of small $\epsilon$ but the toward-better version seems to perform better for finite $\epsilon$.

$$w^{t+1} = w^t + \eta^t \left( \phi(\bar{x}_t, \bar{y}_{\text{direct}}^t) - \phi(\bar{x}_t, \bar{y}_{w^t}(\bar{x}_t)) \right)$$

$$\bar{y}_{\text{direct}}^t = \underset{\tilde{y} \in \mathcal{Y}}{\arg\max} \, (w^t)^\top \phi(\bar{x}_t, \tilde{y}) - \epsilon^t L(\bar{y}, \tilde{y})$$

Our experiments are on the TIMIT speech corpus for which there are published benchmark results [1, 5, 4]. The corpus contains aligned utterances each of which is a pair $(\bar{x}, \bar{y})$ where $\bar{x}$ is a pair of a phonetic sequence and an acoustic sequence and $\bar{y}$ is a desired alignment. We divided the training portion of TIMIT (excluding the SA1 and SA2 utterances) into three disjoint parts containing 1500, 1796, and 100 utterances, respectively. The first part of the training set was used to train a phoneme frame-based classifier, which given a speech frame and a phoneme, outputs the confident that the phoneme was uttered in that frame. The phoneme frame-based classifier is then used as part of a seven dimensional feature map $\phi(\bar{x}, \bar{y}) = \phi((\bar{s}, \bar{p}), \bar{y})$ as described in [5]. The feature set used to train the phoneme classifier consisted of the Mel-Frequency Cepstral Coefficient (MFCC) and the log-energy along with their first and second derivatives ($\Delta + \Delta\Delta$) as described in [5]. The classifier used a Gaussian kernel with $\sigma^2 = 19$ and a trade-off parameter $C = 5.0$. The complete set of 61 TIMIT phoneme symbols were mapped into 39 phoneme symbols as proposed by [6], and was used throughout the training process.

The seven dimensional weight vector $w$ was trained on the second set of 1796 aligned utterances. We trained twice, once for $\tau$-alignment loss and once for $\tau$-insensitive loss, with $\tau = 10$ ms in both cases. Training was done by first setting $w^0 = 0$ and then repeatedly selecting one of the 1796 training pairs at random and performing the update (6) with $\eta^t = 1$ and $\epsilon^t$ set to a fixed value $\epsilon$. It should be noted that if $w^0 = 0$ and $\epsilon^t$ and $\eta^t$ are both held constant at $\epsilon$ and $\eta$ respectively, then the

direction of $w^t$ is independent of the choice of $\eta$. These updates are repeated until the performance of $w^t$ on the third data set (the hold-out set) begins to degrade. This gives a form of regularization known as early stopping. This was repeated for various values of $\epsilon$ and a value of $\epsilon$ was selected based on the resulting performance on the 100 hold-out pairs. We selected $\epsilon = 1.1$ for both loss functions.

We scored the performance of our system on the whole TIMIT test set of 1344 utterances using $\tau$-alignment accuracy (one minus the loss) with $\tau$ set to each of 10, 20, 30 and 40 ms and with $\tau$-insensitive loss with $\tau$ set to 10 ms. As should be expected, for $\tau$ equal to 10 ms the best performance is achieved when the loss used in training matches the loss used in test. Larger values of $\tau$ correspond to a loss function that was not used in training. The results are given in Table 1. We compared our results with [4], which is an HMM/ANN-based system, and with [5], which is based on structural SVM training for $\tau$-insensitive loss. Both systems are considered to be state-of-the-art results on this corpus. As can be seen, our algorithm outperforms the current state-of-the-art results in every tolerance value. Also, as might be expected, the $\tau$-insensitive loss seems more robust to the use of a $\tau$ value at test time that is larger than the $\tau$ value used in training.

## 6  Open Problems and Discussion

The main result of this paper is the loss gradient theorem of Section 3. This theorem provides a theoretical foundation for perceptron-like training methods with updates computed as a difference between the feature vectors of two different inferred outputs where at least one of those outputs is inferred with loss-adjusted inference. Perceptron-like training methods using feature differences between two inferred outputs have already been shown to be successful for machine translation but theoretical justification has been lacking. We also show the value of these training methods in a phonetic alignment problem.

Although we did not give an asymptotic convergence results it should be straightforward to show that under the update given by (6) we have that $w^t$ converges to a local optimum of the objective provided that both $\eta^t$ and $\epsilon^t$ go to zero while $\sum_t \eta^t \epsilon^t$ goes to infinity. For example one could take $\eta^t = \epsilon^t = 1/\sqrt{t}$.

An open problem is how to properly incorporate regularization in the case where only a finite corpus of training data is available. In our phoneme alignment experiments we trained only a seven dimensional weight vector and early stopping was used as regularization. It should be noted that naive regularization with a norm of $w$, such as regularizing with $\lambda ||w||^2$, is nonsensical as the loss $\mathbb{E}\left[L(y, y_w(x))\right]$ is insensitive to the norm of $w$. Regularization is typically done with a surrogate loss function such as hinge loss. Regularization remains an open theoretical issue for direct gradient descent on a desired loss function on a finite training sample. Early stopping may be a viable approach in practice.

Many practical computational problems in areas such as computational linguistics, computer vision, speech recognition, robotics, genomics, and marketing seem best handled by some form of score optimization. In all such applications we have two optimization problems. Inference is an optimization problem (approximately) solved during the operation of the fielded software system. Training involves optimizing the parameters of the scoring function to achieve good performance of the fielded system. We have provided a theoretical foundation for a certain perceptron-like training algorithm by showing that it can be viewed as direct stochastic gradient descent on the loss of the inference system. The main point of this training method is to incorporate domain-specific loss functions, such as the BLEU score in machine translation, directly into the training process with a clear theoretical foundation. Hopefully the theoretical framework provided here will prove helpful in the continued development of improved training methods.

## References

[1] F. Brugnara, D. Falavigna, and M. Omologo. Automatic segmentation and labeling of speech based on hidden markov models. *Speech Communication*, 12:357–370, 1993.

[2] D. Chiang, K. Knight, and W. Wang. 11,001 new features for statistical machine translation. In *Proc. NAACL, 2009*, 2009.

[3] M. Collins. Discriminative training methods for hidden markov models: Theory and experiments with perceptron algorithms. In *Conference on Empirical Methods in Natural Language Processing*, 2002.

[4] J.-P. Hosom. Speaker-independent phoneme alignment using transition-dependent states. *Speech Communication*, 51:352–368, 2009.

[5] J. Keshet, S. Shalev-Shwartz, Y. Singer, and D. Chazan. A large margin algorithm for speech and audio segmentation. *IEEE Trans. on Audio, Speech and Language Processing*, Nov. 2007.

[6] K.-F. Lee and H.-W. Hon. Speaker independent phone recognition using hidden markov models. *IEEE Trans. Acoustic, Speech and Signal Proc.*, 37(2):1641–1648, 1989.

[7] P. Liang, A. Bouchard-Ct, D. Klein, and B. Taskar. An end-to-end discriminative approach to machine translation. In *International Conference on Computational Linguistics and Association for Computational Linguistics (COLING/ACL)*, 2006.

[8] B. Taskar, C. Guestrin, and D. Koller. Max-margin markov networks. In *Advances in Neural Information Processing Systems 17*, 2003.

[9] D.T. Toledano, L.A.H. Gomez, and L.V. Grande. Automatic phoneme segmentation. *IEEE Trans. Speech and Audio Proc.*, 11(6):617–625, 2003.

[10] I. Tsochantaridis, T. Joachims, T. Hofmann, and Y. Altun. Large margin methods for structured and interdependent output variables. *Journal of Machine Learning Research*, 6:1453–1484, 2005.

[11] M. J. Wainwright and M. I. Jordan. Graphical models, exponential families, and variational inference. *Foundations and Trends in Machine Learning*, 1(1-2):1–305, December 2008.

